# Region-based Segmentation and Object Detection

**Stephen Gould**[1]     **Tianshi Gao**[1]     **Daphne Koller**[2]
[1] Department of Electrical Engineering, Stanford University
[2] Department of Computer Science, Stanford University
{sgould,tianshig,koller}@cs.stanford.edu

## Abstract

Object detection and multi-class image segmentation are two closely related tasks that can be greatly improved when solved jointly by feeding information from one task to the other [10, 11]. However, current state-of-the-art models use a separate representation for each task making joint inference clumsy and leaving the classification of many parts of the scene ambiguous.

In this work, we propose a hierarchical region-based approach to joint object detection and image segmentation. Our approach simultaneously reasons about pixels, regions and objects in a coherent probabilistic model. Pixel appearance features allow us to perform well on classifying amorphous background classes, while the explicit representation of regions facilitate the computation of more sophisticated features necessary for object detection. Importantly, our model gives a single unified description of the scene—we explain *every* pixel in the image and enforce global consistency between all random variables in our model.

We run experiments on the challenging Street Scene dataset [2] and show significant improvement over state-of-the-art results for object detection accuracy.

## 1   Introduction

Object detection is one of the great challenges of computer vision, having received continuous attention since the birth of the field. The most common modern approaches scan the image for candidate objects and score each one. This is typified by the sliding-window object detection approach [22, 20, 4], but is also true of most other detection schemes (such as centroid-based methods [13] or boundary edge methods [5]). The most successful approaches combine cues from inside the object boundary (local features) with cues from outside the object (contextual cues), e.g., [9, 20, 6]. Recent works are adopting a more holistic approach by combining the output of multiple vision tasks [10, 11] and are reminiscent of some of the earliest work in computer vision [1]. However, these recent works use a different representation for each subtask, forcing information sharing to be done through awkward feature mappings. Another difficulty with these approaches is that the subtask representations can be inconsistent. For example, a bounding-box based object detector includes many pixels within each candidate detection window that are not part of the object itself. Furthermore, multiple overlapping candidate detections contain many pixels in common. How these pixels should be treated is ambiguous in such approaches. A model that uniquely identifies each pixel is not only more elegant, but is also more likely to produce reliable results since it encodes a bias of the true world (i.e., a visible pixel belongs to only one object).

In this work, we propose a more integrated region-based approach that combines multi-class image segmentation with object detection. Specifically, we propose a hierarchical model that reasons simultaneously about pixels, regions and objects in the image, rather than scanning arbitrary windows. At the region level we label pixels as belonging to one of a number of background classes (currently *sky*, *tree*, *road*, *grass*, *water*, *building*, *mountain*) or a single foreground class. The foreground class is then further classified, at the object level, into one of our known object classes (currently *car* and *pedestrian*) or *unknown*.

Our model builds on the scene decomposition model of Gould et al. [7] which aims to decompose an image into coherent regions by dynamically moving pixel between regions and evaluating these moves relative to a global energy objective. These bottom-up pixel moves result in regions with coherent appearance. Unfortunately, complex objects such as people or cars are composed of several dissimilar regions which will not be combined by this bottom-up approach. Our new hierarchical approach facilitates both bottom-up and top-down reasoning about the scene. For example, we can propose an entire object comprised of multiple regions and evaluate this joint move against our global objective. Thus, our hierarchical model enjoys the best of two worlds: Like multi-class image segmentation, our model uniquely explains every pixel in the image and groups these into semantically coherent regions. Like object detection, our model uses sophisticated shape and appearance features computed over candidate object locations with precise boundaries. Furthermore, our joint model over regions and objects allows context to be encoded through direct semantic relationships (e.g., "car" is usually found on "road").

## 2  Background and Related Work

Our method inherits features from the sliding-window object detector works, such as Torralba et al. [19] and Dalal and Triggs [4], and the multi-class image segmentation work of Shotton et al. [16]. We further incorporate into our model many novel ideas for improving object detection via scene context. The innovative works that inspire ours include predicting camera viewpoint for estimating the real world size of object candidates [12], relating "things" (objects) to nearby "stuff" (regions) [9], co-occurrence of object classes [15], and general scene "gist" [18].

Recent works go beyond simple appearance-based context and show that holistic scene understanding (both geometric [11] and more general [10]) can significantly improve performance by combining related tasks. These works use the output of one task (e.g., object detection) to provide features for other related tasks (e.g., depth perception). While they are appealing in their simplicity, current models are not tightly coupled and may result in incoherent outputs (e.g., the pixels in a bounding box identified as "car" by the object detector, may be labeled as "sky" by an image segmentation task). In our method, all tasks use the same region-based representation which forces consistency between variables. Intuitively this leads to more robust predictions.

The decomposition of a scene into regions to provide the basis for vision tasks exists in some scene parsing works. Notably, Tu et al. [21] describe an approach for identifying regions in the scene. Their approach has only be shown to be effective on text and faces, leaving much of the image unexplained. Sudderth et al. [17] relate scenes, objects and parts in a single hierarchical framework, but do not provide an exact segmentation of the image. Gould et al. [7] provides a complete description of the scene using dynamically evolving decompositions that explain every pixel (both semantically and geometrically). However, the method cannot distinguish between between foreground objects and often leaves them segmented into multiple dissimilar pieces. Our work builds on this approach with the aim of classifying objects.

Other works attempt to integrate tasks such as object detection and multi-class image segmentation into a single CRF model. However, these models either use a different representation for object and non-object regions [23] or rely on a pixel-level representation [16]. The former does not enforce label consistency between object bounding boxes and the underlying pixels while the latter does not distinguish between adjacent objects of the same class.

Recent work by Gu et al. [8] also use regions for object detection instead of the traditional sliding-window approach. However, unlike our method, they use a single over-segmentation of the image and make the strong assumption that each segment represents a (probabilistically) recognizable object part. Our method, on the other hand, assembles objects (and background regions) using segments from multiple different over-segmentations. The multiple over-segmentations avoids errors made by any one segmentation. Furthermore, we incorporate background regions which allows us to eliminate large portions of the image thereby reducing the number of component regions that need to be considered for each object.

Liu et al. [14] use a non-parametric approach to image labeling by warping a given image onto a large set of labeled images and then combining the results. This is a very effective approach since it scales easily to a large number of classes. However, the method does not attempt to understand the scene semantics. In particular, their method is unable to break the scene into separate objects (e.g., a row of cars will be parsed as a single region) and cannot capture combinations of classes not present in the training set. As a result, the approach performs poorly on most foreground object classes.

# 3   Region-based Model for Object Detection

We now present an overview of our joint object detection and scene segmentation model. This model combines scene structure and semantics in a coherent energy function.

## 3.1   Energy Function

Our model builds on the work of Gould et al. [7] which aims to decompose a scene into a number ($K$) of semantically consistent regions. In that work, each pixel $p$ in the image $\mathcal{I}$ belongs to exactly one region, identified by its region-correspondence variable $R_p \in \{1, \ldots, K\}$. The $r$-th region is then simply the set of pixels $\mathcal{P}_r$ whose region-correspondence variable equals $r$, i.e., $\mathcal{P}_r = \{p : R_p = r\}$. In our notation we will always use $p$ and $q$ to denote pixels, $r$ and $s$ to denote regions, and $o$ to denote objects. Double indices indicate pairwise terms between adjacent entities (e.g., $pq$ or $rs$).

Regions, while visually coherent, may not encompass entire objects. Indeed, in the work of Gould et al. [7] foreground objects tended to be over-segmented into multiple regions. We address this deficiency by allowing an object to be composed of many regions (rather than trying to force dissimilar regions to merge). The object to which a region belongs is denoted by its object-correspondence variable $O_r \in \{\varnothing, 1, \ldots, N\}$. Some regions, such as background, do not belong to any object which we denote by $O_r = \varnothing$. Like regions, the set of pixels that comprise the $o$-th object is denoted by $\mathcal{P}_o = \bigcup_{r:O_r=o} \mathcal{P}_r$. Currently, we do not allow a single region or object to be composed of multiple disconnected components.

Random variables are associated with the various entities (pixels, regions and objects) in our model. Each pixel has a local appearance feature vector $\alpha_p \in \mathbb{R}^n$ (see [7]). Each region has an appearance variable $A_r$ that summarizes the appearance of the region as a whole, a semantic class label $S_r$ (such as "road" or "foreground object"), and an object-correspondence variable $O_r$. Each object, in turn, has an associated object class label $C_o$ (such as "car" or "pedestrian"). The final component in our model is the horizon which captures global geometry information. We assume that the image was taken by a camera with horizontal axis parallel to the ground and model the horizon $v^{\text{hz}} \in [0, 1]$ as the normalized row in the image corresponding to its location. We quantize $v^{\text{hz}}$ into the same number of rows as the image.

We combine the variables in our model into a single coherent energy function that captures the structure and semantics of the scene. The energy function includes terms for modeling the location of the horizon, region label preferences, region boundary quality, object labels, and contextual relationships between objects and regions. These terms are described in detail below. The combined energy function $E(\mathbf{R}, \mathbf{S}, \mathbf{O}, \mathbf{C}, v^{\text{hz}} \mid \mathcal{I}, \boldsymbol{\theta})$ has the form:

$$E = \psi^{\text{hz}}(v^{\text{hz}}) + \sum_r \psi_r^{\text{reg}}(S_r, v^{\text{hz}}) + \sum_{r,s} \psi_{rs}^{\text{bdry}} + \sum_o \psi_o^{\text{obj}}(C_o, v^{\text{hz}}) + \sum_{o,r} \psi_{or}^{\text{ctxt}}(C_o, S_r) \quad (1)$$

where for notational clarity the subscripts on the factors indicate that they are functions of the pixels (appearance and shape) belonging to the regions, i.e., $\psi_r^{\text{reg}}$ is also a function of $\mathcal{P}_r$, etc. It is assumed that all terms are conditioned on the observed image $\mathcal{I}$ and model parameters $\boldsymbol{\theta}$. The summation over context terms includes all ordered pairs of adjacent objects and regions, while the summation over boundary terms is over unordered pairs of regions. An illustration of the variables in the energy function is shown in Figure 1.

The first three energy terms are adapted from the model of [7]. We briefly review them here:

**Horizon term.** The $\psi^{\text{hz}}$ term captures the a priori location of the horizon in the scene and, in our model, is implemented as a log-gaussian $\psi^{\text{hz}}(v^{\text{hz}}) = -\log \mathcal{N}(v^{\text{hz}}; \mu, \sigma^2)$ with parameters $\mu$ and $\sigma$ learned from labeled training images.

Knowing the location of the horizon allows us to compute the world height of an object in the scene. Using the derivation from Hoiem et al. [12], it can be shown that the height $y_k$ of an object (or region) in the scene can be approximated as $y_k \approx h \frac{v_t - v_b}{v^{\text{hz}} - v_b}$ where $h$ is the height of the camera origin above the ground, and $v_t$ and $v_b$ are the row of the top-most and bottom-most pixels in the object/region, respectively. In our current work, we assume that all images were taken from the same height above the ground, allowing us to use $\frac{v_t - v_b}{v^{\text{hz}} - v_b}$ as a feature in our region and object terms.

**Region term.** The region term $\psi^{\text{reg}}$ in our energy function captures the preference for a region to be assigned different semantic labels (currently *sky*, *tree*, *road*, *grass*, *water*, *building*, *mountain*, *foreground*). For convenience we include the $v^{\text{hz}}$ variable in this term to provide rough geometry information. If a region is associated with an object, then we constrain the assignment of its class label to *foreground* (e.g., a "sky" region cannot be part of a "car" object).

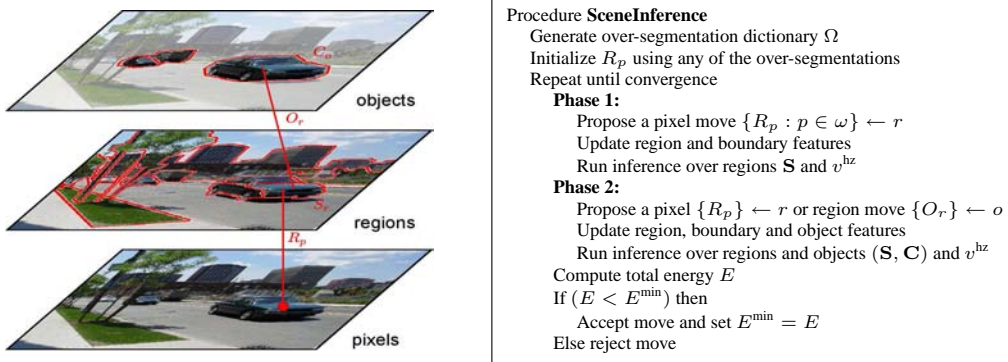

```
Procedure SceneInference
    Generate over-segmentation dictionary Ω
    Initialize R_p using any of the over-segmentations
    Repeat until convergence
        Phase 1:
            Propose a pixel move {R_p : p ∈ ω} ← r
            Update region and boundary features
            Run inference over regions S and v^hz
        Phase 2:
            Propose a pixel {R_p} ← r or region move {O_r} ← o
            Update region, boundary and object features
            Run inference over regions and objects (S, C) and v^hz
    Compute total energy E
    If (E < E^min) then
        Accept move and set E^min = E
    Else reject move
```

**Figure 1:** Illustration of the entities in our model (left) and inference algorithm (right). See text for details.

More formally, let $N_r$ be the number of pixels in region $r$, i.e., $N_r = \sum_p \mathbf{1}\{R_p = r\}$, and let $\phi_r : \left(\mathcal{P}_r, v^{\text{hz}}, \mathcal{I}\right) \mapsto \mathbb{R}^n$ denote the features for the $r$-th region. The region term is then

$$\psi_r^{\text{reg}}(S_r, v^{\text{hz}}) = \begin{cases} \infty & \text{if } O_r \neq \varnothing \text{ and } S_r \neq foreground \\ -\eta^{\text{reg}} N_r \log \sigma\left(S_r \mid \phi_r; \theta^{\text{reg}}\right) & \text{otherwise} \end{cases} \tag{2}$$

where $\sigma(\cdot)$ is the multi-class logit $\sigma(y \mid x; \theta) = \dfrac{\exp\{\theta_y^T x\}}{\sum_{y'} \exp\{\theta_{y'}^T x\}}$ and $\eta^{\text{reg}}$ is the relative weight of the region term versus the other terms in the model.

**Boundary term.** The term $\psi^{\text{bdry}}$ penalizes two adjacent regions with similar appearance or lack of boundary contrast. This helps to merge coherent pixels into a single region. We combine two metrics in this term: the first captures region similarity as a whole, the second captures contrast along the common boundary between the regions. Specifically, let $d\left(x, y; S\right) = \sqrt{(x-y)^T S^{-1} (x-y)}$ denote the Mahalanobis distance between vectors $x$ and $y$, and $\mathcal{E}_{rs}$ be the set of pixels along the boundary. Then the boundary term is

$$\psi_{rs}^{\text{bdry}} = \eta_A^{\text{bdry}} \cdot |\mathcal{E}_{rs}| \cdot e^{-\frac{1}{2} d(A_r, A_s; \Sigma_A)^2} + \eta_\alpha^{\text{bdry}} \sum_{(p,q) \in \mathcal{E}_{rs}} e^{-\frac{1}{2} d(\alpha_p, \alpha_q; \Sigma_\alpha)^2} \tag{3}$$

where the $\Sigma_A$ and $\Sigma_\alpha$ are the image-specific pixel appearance covariance matrix computed over all pixels and neighboring pixels, respectively. In our experiments we restrict $\Sigma_A$ to be diagonal and set $\Sigma_\alpha = \beta \mathbf{I}$ with $\beta = \mathbf{E}\left[\|\alpha_p - \alpha_q\|^2\right]$ as in Shotton et al. [16]. The parameters $\eta_A^{\text{bdry}}$ and $\eta_\alpha^{\text{bdry}}$ encode the trade-off between the region similarity and boundary contrast terms and weight them against the other terms in the energy function (Equation 1).

Note that the boundary term does not include semantic class or object information. The term purely captures segmentation coherence in terms of appearance.

**Object term.** Going beyond the model in [7], we include object terms $\psi^{\text{obj}}$ in our energy function that score the likelihood of a group of regions being assigned a given object label. We currently classify objects as either *car*, *pedestrian* or *unknown*. The *unknown* class includes objects like trash cans, street signs, telegraph poles, traffic cones, bicycles, etc. Like the region term, the object term is defined by a logistic function that maps object features $\phi_o : \left(\mathcal{P}_o, v^{\text{hz}}, \mathcal{I}\right) \mapsto \mathbb{R}^n$ to probability of each object class. However, since our region layer already identifies foreground regions, we would like our energy to improve only when we recognize known object classes. We therefore bias the object term to give zero contribution to the energy for the class *unknown*.[1] Formally we have

$$\psi_n^{\text{obj}}(C_o, v^{\text{hz}}) = -\eta^{\text{obj}} N_o \left(\log \sigma\left(C_o \mid \phi_o; \theta^{\text{obj}}\right) - \log \sigma\left(unknown \mid \phi_o; \theta^{\text{obj}}\right)\right) \tag{4}$$

where $N_o$ is the number of pixels belonging to the object.

**Context term.** Intuitively, contextual information which relates objects to their local background can improve object detection. For example, Heitz and Koller [9] showed that detection rates improve by relating "things" (objects) to "stuff" (background). Our model has a very natural way of

encoding such relationships through pairwise energy terms between objects $C_o$ and regions $S_r$. We do not encode contextual relationships between region classes (i.e., $S_r$ and $S_s$) since these rarely help.[2] Contextual relationships between foreground objects (i.e., $C_o$ and $C_m$) may be beneficial (e.g., people found on bicycles), but are not considered in this work. Formally, the context term is

$$\psi_{or}^{\text{ctxt}}(C_o, S_r) = -\eta^{\text{ctxt}} \log \sigma \left( C_o \times S_r \mid \phi_{or}; \theta^{\text{ctxt}} \right) \qquad (5)$$

where $\phi_{or} : (\mathcal{P}_o, \mathcal{P}_r, \mathcal{I}) \mapsto \mathbb{R}^n$ is a pairwise feature vector for object $o$ and region $r$, $\sigma(\cdot)$ is the multi-class logit, and $\eta^{\text{ctxt}}$ weights the strength of the context term relative to other terms in the energy function. Since the pairwise context term is between objects and (background) regions it grows linearly with the number of object classes. This has a distinct advantage over approaches which include a pairwise term between all classes resulting in quadratic growth.

## 3.2 Object Detectors

Performing well at object detection requires more than simple region appearance features. Indeed, the power of state-of-the-art object detectors is their ability to model localized appearance and general shape characteristics of an object class. Thus, in addition to raw appearance features, we append to our object feature vector $\phi_o$ features derived from such object detection models. We discuss two methods for adapting state-of-the-art object detector technologies for this purpose.

In the first approach, we treat the object detector as a black-box that returns a score per (rectangular) candidate window. However, recall that an object in our model is defined by a contiguous set of pixels $\mathcal{P}_o$, not a rectangular window. In the black-box approach, we naively place a bounding box (at the correct aspect ratio) around these pixels and classify the entire contents of the box. To make classification more robust we search candidate windows in a small neighborhood (defined over scale and position) around this bounding box, and take as our feature the output of highest scoring window. In our experiments we test this approach using the HOG detector of Dalal and Triggs [4] which learns a linear SVM classifier over feature vectors constructed by computing histograms of gradient orientations in fixed-size overlapping cells within the candidate window.

Note that in the above black-box approach many of the pixels within the bounding box are not actually part of the object (consider, for example, an L-shaped region). A better approach is to mask out all pixels not belonging to the object. In our implementation, we use a soft mask that attenuates the intensity of pixels outside the object based on their distance to the object boundary (see Figure 2). This has the dual advantage of preventing hard edge artifacts and being less sensitive to segmentation errors. The masked window is used at both training and test time. In our experiments we test this more integrated approach using the patch-based features of Torralba *et al.* [19, 20]. Here features are extracted by matching small rectangular patches at various locations within the masked window and combining these weak responses using boosting. Object appearance and shape are captured by operating on both the original (intensity) image and the edge-filtered image.

For both approaches, we append the score (for each object) from the object detection classifiers—linear SVM or boosted decision trees—to the object feature vector $\phi_o$.

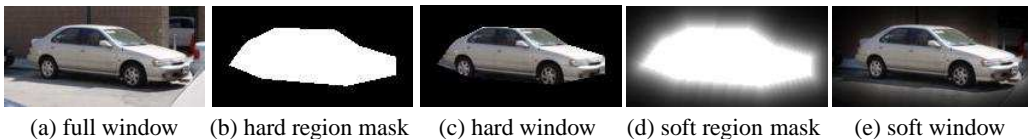

| (a) full window | (b) hard region mask | (c) hard window | (d) soft region mask | (e) soft window |

**Figure 2:** Illustration of soft mask for proposed object regions.

An important parameter for sliding-window detectors is the base scale at which features are extracted. Scale-invariance is achieved by successively down-sampling the image. Below the base-scale, feature matching becomes inaccurate, so most detectors will only find objects above some minimum size. Clearly there exists a trade-off between the desire to detect small objects, feature quality, and computational cost. To reduce the computational burden of running our model on high-resolution images while still being able to identify small objects, we employ a multi-scale approach. Here we run our scene decomposition algorithm on a low-resolution ($320 \times 240$) version of the scene, but extract features from the original high-resolution version. That is, when we extract object-detector features we map the object pixels $\mathcal{P}_o$ onto the original image and extract our features at the higher resolution.

# 4 Inference and Learning

We now describe how we perform inference and learn the parameters of our energy function.

## 4.1 Inference

We use a modified version of the hill-climbing inference algorithm described in Gould et al. [7], which uses multiple over-segmentations to propose large moves in the energy space. An overview of this procedure is shown in the right of Figure 1. We initialize the scene by segmenting the image using an off-the-shelf unsupervised segmentation algorithm (in our experiments we use mean-shift [3]). We then run inference using a two-phased approach.

In the first phase, we want to build up a good set of initial regions before trying to classify them as objects. Thus we remove the object variables $\mathbf{O}$ and $\mathbf{C}$ from the model and artificially increase the boundary term weights ($\eta_\alpha^{\text{bdry}}$ and $\eta_A^{\text{bdry}}$) to promote merging. In this phase, the algorithm behaves exactly as in [7] by iteratively proposing re-assignments of pixels to regions (variables $\mathbf{R}$) and re-computes the optimal assignment to the remaining variables ($\mathbf{S}$ and $v^{\text{hz}}$). If the overall energy for the new configuration is lower, the move is accepted, otherwise the previous configuration is restored and the algorithm proposes a different move. The algorithm proceeds until no further reduction in energy can be found after exhausting all proposal moves from a pre-defined set (see Section 4.2).

In the second phase, we anneal the boundary term weights and introduce object variables over all foreground regions. We then iteratively propose merges and splits of objects (variables $\mathbf{O}$) as well as high-level proposals (see Section 4.2 below) of new regions generated from sliding-window object candidates (affecting both $\mathbf{R}$ and $\mathbf{O}$). After a move is proposed, we recompute the optimal assignment to the remaining variables ($\mathbf{S}$, $\mathbf{C}$ and $v^{\text{hz}}$). Again, this process repeats until the energy cannot be reduced by any of the proposal moves.

Since only part of the scene is changing during any iteration we only need to recompute the features and energy terms for the regions affected by a move. However, inference is still slow given the sophisticated features that need to be computed and the large number of moves considered. To improve running time, we leave the context terms $\psi^{\text{ctxt}}$ out of the model until the last iteration through the proposal moves. This allows us to maximize each region term independently during each proposal step—we use an iterated conditional modes (ICM) update to optimize $v^{\text{hz}}$ after the region labels have been inferred. After introducing the context term, we use max-product belief propagation to infer the optimal joint assignment to $\mathbf{S}$ and $\mathbf{C}$. Using this approach we can process an image in under five minutes.

## 4.2 Proposal Moves

We now describe the set of pixel and region proposal moves considered by our algorithm. These moves are relative to the current best scene decomposition and are designed to take large steps in the energy space to avoid local minima. As discussed above, each move is accepted if it results in a lower overall energy after inferring the optimal assignment for the remaining variables.

The main set of pixel moves are described in [7] but briefly repeated here for completeness. The most basic move is to merge two adjacent regions. More sophisticated moves involve local re-assignment of pixels to neighboring regions. These moves are proposed from a pre-computed dictionary of image segments $\Omega$. The dictionary is generated by varying the parameters of an unsupervised over-segmentation algorithm (in our case mean-shift [3]) and adding each segment $\omega$ to the dictionary. During inference, these segments are used to propose a re-assignment of all pixels in the segment to a neighboring region or creation of new region. These bottom-up proposal moves work well for background classes, but tend to result in over-segmented foreground classes which have heterogeneous appearance, for example, one would not expect the wheels and body of a car to be grouped together by a bottom-up approach.

An analogous set of moves can be used for merging two adjacent objects or assigning regions to objects. However, if an object is decomposed into multiple regions, this bottom-up approach is problematic as multiple such moves may be required to produce a complete object. When performed independently, these moves are unlikely to improve the energy. We get around this difficulty by introducing a new set of powerful top-down proposal moves based on object detection candidates. Here we use pre-computed candidates from a sliding-window detector to propose new foreground regions with corresponding object variable. Instead of proposing the entire bounding-box from the detector, we propose the set of intersecting segments (from our segmentation dictionary $\Omega$) that are fully contained within the bounding-box in a single move.

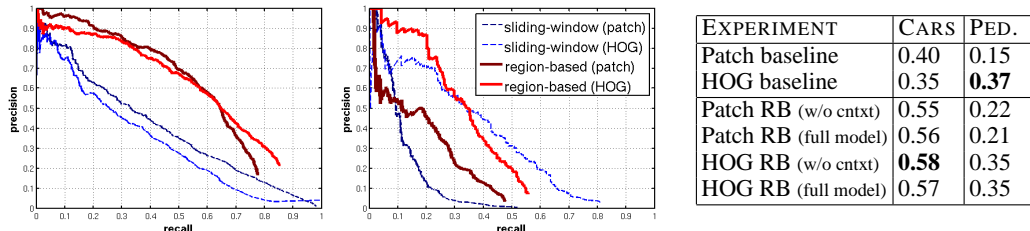

**Figure 3:** PR curves for car (left) and pedestrian (right) detection on the Street Scene dataset [2]. The table shows 11-pt average precision for variants of the baseline sliding-window and our region-based (RB) approach.

| EXPERIMENT | CARS | PED. |
|---|---|---|
| Patch baseline | 0.40 | 0.15 |
| HOG baseline | 0.35 | **0.37** |
| Patch RB (w/o cntxt) | 0.55 | 0.22 |
| Patch RB (full model) | 0.56 | 0.21 |
| HOG RB (w/o cntxt) | **0.58** | 0.35 |
| HOG RB (full model) | 0.57 | 0.35 |

### 4.3 Learning

We learn the parameters of our model from labeled training data in a piecewise fashion. First, the individual terms are learned using the maximum-likelihood objective for the subset of variables within each term. The relative weights ($\eta^{\text{reg}}$, $\eta^{\text{obj}}$, etc.) between the terms are learned through cross-validation on a subset of the training data. Boosted pixel appearance features (see [7]) and object detectors are learned separately and their output provided as input features to the combined model.

For both the base object detectors and the parameters of the region and object terms, we use a closed-loop learning technique where we first learn an initial set of parameters from training data. We then run inference on our training set and record mistakes made by the algorithm (false-positives for object detection and incorrect moves for the full algorithm). We augment the training data with these mistakes and re-train. This process gives a significant improvement to the final results.

## 5 Experiments

We conduct experiments on the challenging Street Scene dataset [2]. This is a dataset consisting of 3547 high-resolution images of urban environments. We rescaled the images to $320 \times 240$ before running our algorithm. The dataset comes with hand-annotated region labels and object boundaries. However, the annotations use rough overlapping polygons, so we used Amazon's Mechanical Turk to improve the labeling of the background classes only. We kept the original object polygons to be consistent with other results on this dataset.

We divided the dataset into five folds—the first fold (710 images) was used for testing and the remaining four used for training. The multi-class image segmentation component of our model achieves an overall pixel-level accuracy of 84.2% across the eight semantic classes compared to 83.0% for the pixel-based baseline method described in [7]. More interesting was our object detection performance. The test set contained 1183 cars and 293 pedestrians with average size of $86 \times 48$ and $22 \times 49$ pixels, respectively. Many objects are occluded making this a very difficult dataset.

Since our algorithm produces MAP estimation for the scene we cannot simply generate a precision-recall curve by varying the object classifier threshold as is usual for reporting object detection results. Instead we take the max-marginals for each $C_n$ variable at convergence of our algorithm and sweep over thresholds for each object separately to generate a curve. An attractive aspect of this approach is that our method does not have overlapping candidates and hence does not require arbitrary post-processing such as non-maximal suppression of sliding-window detections.

Our results are shown in Figure 3. We also include a comparison to two baseline sliding-window approaches. Our method significantly improves over the baselines for car detection. For pedestrian detection, our method shows comparable performance to the HOG baseline which has been specifically engineered for this task. Notice that our method does not achieve 100% recall (even at low precision) due to the curves being generated from the MAP assignment in which pixels have already been grouped into regions. Unlike the baselines, this forces only one candidate object per region. However, by trading-off the strength (and hence operating point) of the energy terms in our model we can increase the maximum recall for a given object class (e.g., by increasing the weight of the object term by a factor of 30 we were able to increase pedestrian recall from 0.556 to 0.673).

Removing the pairwise context term does not have a significant affect on our results. This is due to the encoding of semantic context through the region term and the fact that all images were of urban scenes. However, we believe that on a dataset with more varied backgrounds (e.g., rural scenes) context would play a more important role.

We show some example output from our algorithm in Figure 4. The first row shows the original image (left) together with annotated regions and objects (middle-left), regions (middle-right) and predicted horizon (right). Notice how multiple regions get grouped together into a single object. The remaining rows show a selection of results (image and annotated output) from our method.

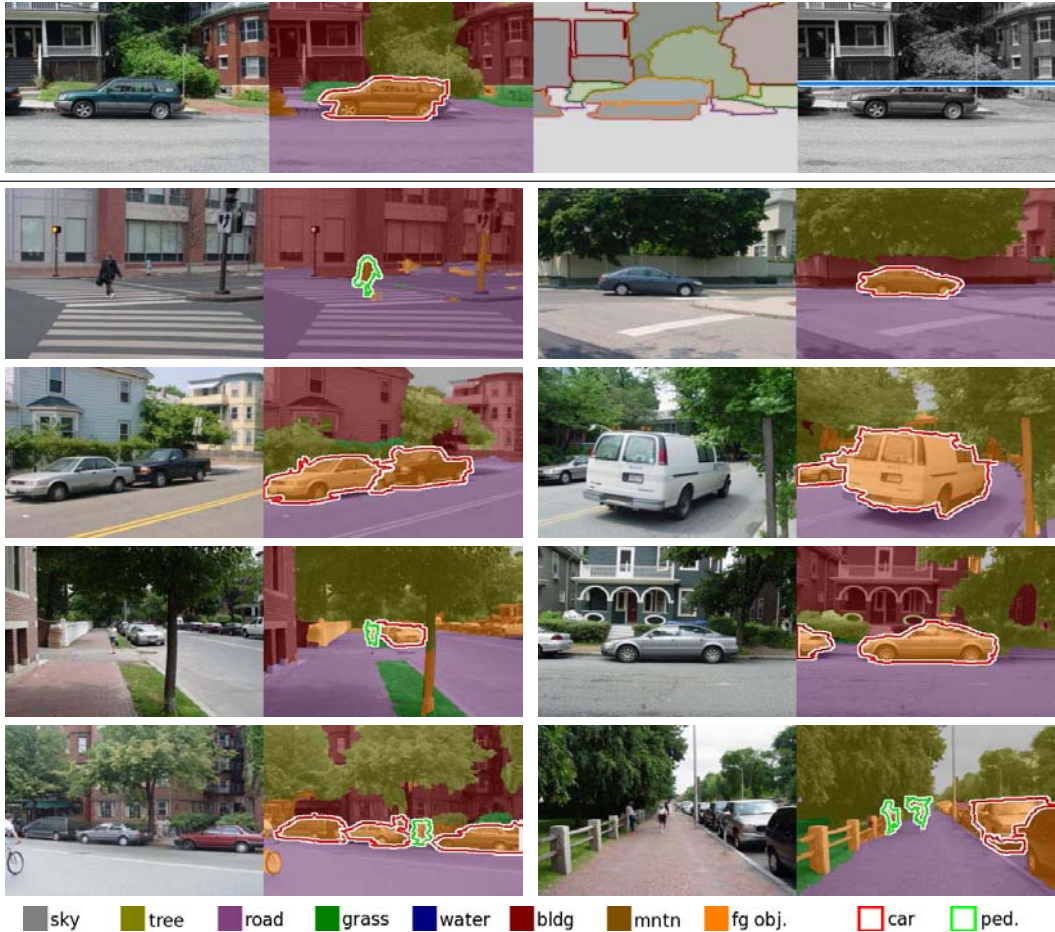

**Figure 4:** Qualitative results from our experiments. Top row shows original image, annotated regions and objects, region boundaries, and predicted horizon. Other examples show original image (left) and overlay colored by semantic class and detected objects (right).

## 6 Discussion

In this paper we have presented a hierarchical model for joint object detection and image segmentation. Our novel approach overcomes many of the problems associated with trying to combine related vision tasks. Importantly, our method explains every pixel in the image and enforces consistency between random variables from different tasks. Furthermore, our model is encapsulated in a modular energy function which can be easily analyzed and improved as new computer vision technologies become available.

One of the difficulties in our model is learning the trade-off between energy terms—too strong a boundary penalty and all regions will be merged together, while too weak a penalty and the scene will be split into too many segments. We found that a closed-loop learning regime where mistakes from running inference on the training set are used to increase the diversity of training examples made a big difference to performance.

Our work suggests a number of interesting directions for future work. First, our greedy inference procedure can be replaced with a more sophisticated approach that makes more global steps. More importantly, our region-based model has the potential for providing holistic unified understanding of an entire scene. This has the benefit of eliminating many of the implausible hypotheses that plague current computer vision algorithms. Furthermore, by clearly delineating what is recognized, our framework directly present hypotheses for objects that are currently unknown providing the potential for increasing our library of characterized objects using a combination of supervised and unsupervised techniques.

**Acknowledgments.** This work was supported by the NSF under grant IIS 0917151, MURI contract N000140710747, and The Boeing Company. We thank Pawan Kumar and Ben Packer for helpful discussions.

## Footnotes

[1]This results in the technical condition of allowing $O_r$ to take the value $\varnothing$ for *unknown* foreground regions without affecting the energy.

[2]The most informative region-to-region relationship is that *sky* tends to be above ground (*road*, *grass*, or *water*). This information is already captured by including the horizon in our region term.

# References

[1] H.G. Barrow and J.M. Tenenbaum. Computational vision. *IEEE*, 1981.

[2] S. Bileschi and L. Wolf. A unified system for object detection, texture recognition, and context analysis based on the standard model feature set. In *BMVC*, 2005.

[3] D. Comaniciu and P. Meer. Mean shift: A robust approach toward feature space analysis. *PAMI*, 2002.

[4] N. Dalal and B. Triggs. Histograms of oriented gradients for human detection. In *CVPR*, 2005.

[5] V. Ferrari, L. Fevrier, F. Jurie, and C. Schmid. Groups of adjacent contour segments for object detection. *PAMI*, 2008.

[6] M. Fink and P. Perona. Mutual boosting for contextual inference. In *NIPS*, 2003.

[7] Stephen Gould, Rick Fulton, and Daphne Koller. Decompsing a scene into geometric and semantically consistent regions. In *ICCV*, 2009.

[8] C. Gu, J. J. Lim, P. Arbelaez, and J. Malik. Recognition using regions. In *CVPR*, 2009.

[9] G. Heitz and D. Koller. Learning spatial context: Using stuff to find things. In *ECCV*, 2008.

[10] G. Heitz, S. Gould, A. Saxena, and D. Koller. Cascaded classification models: Combining models for holistic scene understanding. In *NIPS*, 2008.

[11] D. Hoiem, A. A. Efros, and M. Hebert. Closing the loop on scene interpretation. *CVPR*, 2008.

[12] D. Hoiem, A. A. Efros, and M. Hebert. Putting objects in perspective. *IJCV*, 2008.

[13] B. Leibe, A. Leonardis, and B. Schiele. Combined object categorization and segmentation with an implicit shape model. In *ECCV*, 2004.

[14] C. Liu, J. Yuen, and A. Torralba. Nonparametric scene parsing: Label transfer via dense scene alignment. In *CVPR*, 2009.

[15] A. Rabinovich, A. Vedaldi, C. Galleguillos, E. Wiewiora, and S. Belongie. Objects in context. In *ICCV*, 2007.

[16] J. Shotton, J. Winn, C. Rother, and A. Criminisi. TextonBoost: Joint appearance, shape and context modeling for multi-class object recognition and segmentation. In *ECCV*, 2006.

[17] E. Sudderth, A. Torralba, W. Freeman, and A. Willsky. Describing visual scenes using transformed objects and parts. In *IJCV*, 2007.

[18] A. Torralba, K. P. Murphy, W. T. Freeman, and M. A. Rubin. Context-based vision system for place and object recognition, 2003.

[19] A. Torralba, K. Murphy, and W. Freeman. Sharing features: efficient boosting procedures for multiclass object detection. In *CVPR*, 2004.

[20] A. Torralba, K. Murphy, and W. Freeman. Contextual models for object detection using boosted random fields. In *NIPS*, 2004.

[21] Z. Tu, X. Chen, A. L. Yuille, and S.-C. Zhu. Image parsing: Unifying segmentation, detection, and recognition. In *ICCV*, 2003.

[22] P. Viola and M. J. Jones. Robust real-time face detection. *IJCV*, 2004.

[23] C. Wojek and B. Schiele. A dynamic conditional random field model for joint labeling of object and scene classes. In *ECCV*, 2008.

